# Learning Control Under Extreme Uncertainty

Vijaykumar Gullapalli
Computer Science Department
University of Massachusetts
Amherst, MA 01003

## Abstract

A peg-in-hole insertion task is used as an example to illustrate the utility of direct associative reinforcement learning methods for learning control under real-world conditions of uncertainty and noise. Task complexity due to the use of an unchamfered hole and a clearance of less than $0.2mm$ is compounded by the presence of positional uncertainty of magnitude exceeding 10 to 50 times the clearance. Despite this extreme degree of uncertainty, our results indicate that direct reinforcement learning can be used to learn a robust reactive control strategy that results in skillful peg-in-hole insertions.

## 1 INTRODUCTION

Many control tasks of interest today involve controlling complex nonlinear systems under uncertainty and noise.[1] Because traditional control design techniques are not very effective under such circumstances, methods for learning control are becoming increasingly popular. Unfortunately, in many of these control tasks, it is difficult to obtain training information in the form of prespecified instructions on how to perform the task. Therefore supervised learning methods are not directly applicable. At the same time, evaluating the performance of a controller on the task is usually fairly straightforward, and hence these tasks are ideally suited for the application of *associative reinforcement learning* (Barto & Anandan, 1985).

In associative reinforcement learning, the learning system's interactions with its environment are evaluated by a critic, and the goal of the learning system is to learn to respond to each input with the action that has the best expected evaluation. In learning control tasks, the learning system is the controller, its actions are control signals, and the critic's evaluations are based on the performance criterion associated with the control task. Two kinds of associative reinforcement learning methods, direct and indirect, can be distinguished (e.g., Gullapalli, 1992). Indirect reinforcement learning methods construct and use a model of the environment and the critic (modeled either separately or together), while direct reinforcement learning methods do not.

We have previously argued (Gullapalli, 1992; Barto & Gullapalli, 1992) that in the presence of uncertainty, hand-crafting or learning an adequate model—imperative if one is to use indirect methods for training the controller—can be very difficult. Therefore, it can be expeditious to use direct reinforcement learning methods in such situations. In this paper, a peg-in-hole insertion task is used as an example to illustrate the utility of direct associative reinforcement learning methods for learning control under real-world conditions of uncertainty.

## 2    PEG-IN-HOLE INSERTION

Peg-in-hole insertion has been widely used by roboticists for testing various approaches to robot control and has also been studied as a canonical robot assembly operation (Whitney, 1982; Gustavson, 1984; Gordon, 1986). Although the abstract peg-in-hole task can be solved quite easily, real-world conditions of uncertainty due to (1) errors and noise in sensory feedback, (2) errors in execution of motion commands, and (3) uncertainty due to movement of the part grasped by the robot can substantially degrade the performance of traditional control methods. Approaches proposed for peg-in-hole insertion under uncertainty can be grouped into two major classes: methods based on off-line planning, and methods based on reactive control.

Off-line planning methods combine geometric analysis of the peg-hole configuration with analysis of the task statics to determine motion strategies that will result in successful insertion (Whitney, 1982; Gustavson, 1984; Gordon, 1986). In the presence of uncertainty in sensing and control, researchers have suggested incorporating the uncertainty into the geometric model of the task in configuration space (e.g., Lozano-Perez et al., 1984; Erdmann, 1986; Caine et al., 1989; Donald, 1986). Off-line planning is based on the assumption that a realistic characterization of the margins of uncertainty is available, which is a strong assumption when dealing with real-world systems.

Methods based on reactive control, in comparison, try to counter the effects of uncertainty with on-line modification of the motion control based on sensory feedback. Often, *compliant* motion control is used, in which the trajectory is modified by contact forces or tactile stimuli occurring during the motion. The compliant behavior either is actively generated or occurs passively due to the physical characteristics of the robot (Whitney, 1982; Asada, 1990). However, as Asada (1990) points out, many tasks including the peg insertion task require complex nonlinear compliance or admittance behavior that is beyond the capability of a passive mechanism. Unfortunately, humans find it quite difficult to prespecify appropri-

ate compliant behavior (Lozano-Perez et al., 1984), especially in the presence of uncertainty. Hence techniques for learning compliant behavior can be very useful.

We demonstrate our approach to learning a reactive control strategy for peg-in-hole insertion by training a controller to perform peg-in-hole insertions using a Zebra Zero robot. The Zebra Zero is equipped with joint position encoders and a six-axis force sensor at its wrist, whose outputs are all subject to uncertainty. Before describing the controller and presenting its performance in peg insertion, we present some experimental data quantifying the uncertainty in position and force sensors.

## 3    QUANTIFYING THE SENSOR UNCERTAINTY

In order to quantify the position uncertainty under varying load conditions similar to those that occur when the peg is interacting with the hole, we compared the sensed peg position with its actual position in cartesian space under different load conditions. In one such experiment, the robot was commanded to maintain a fixed position under five different loads conditions applied sequentially: no load, and a fixed load of 0.12Kgf applied in the $\pm x$ and $\pm y$ directions. Under each condition, the position and force feedback from the robot sensors, as well as the actual $x$-$y$ position of the peg were recorded.

The sensed and actual $x$-$y$ positions of the peg are shown in Table 1. The sensed $x$-$y$ positions were computed from the joint positions sensed by the Zero's joint position encoders. As can be seen from the table, there is a large discrepancy between the sensed and actual positions of the peg: while the actual change in the peg's position under the external load was of the order of 2 to 3$mm$, the largest sensed change in position was less than 0.025$mm$. In comparison, the clearance between the peg and the hole (in the 3D task) was 0.175$mm$. From observations of the robot, we could determine that the uncertainty in position was primarily due to gear backlash. Other factors affecting the uncertainty include the posture of the robot arm, which affects the way the backlash is loaded, and interactions between the peg and the environment.

Table 1: Sensed And Actual Positions Under 5 Different Load Conditions

| Load Condition | Sensed $x$-$y$ Position ($mm$) | Actual $x$-$y$ Position ($mm$) |
|---|---|---|
| No load position | (0.0, 0.000000) | (0.0, 0.0) |
| With $-y$ load | (0.0, $-$0.014673) | (0.0, $-$2.5) |
| With $+x$ load | (0.0, 0.000000) | (1.9, $-$0.3) |
| With $+y$ load | (0.0, 0.024646) | ($-$2.9, $-$0.2) |
| With $-x$ load | (0.0, 0.010026) | (0.3, 2.2) |
| Final (no load) position | (0.0, 0.000000) | (0.0, $-$0.6) |

Figure 1 shows 30 time-step samples of the force sensor output for each of the load conditions described above. As can be seen from the figure, there is considerable sensor noise, especially in recording moments. Although designing a controller that can robustly perform peg insertions despite the large uncertainty in sensory input

is difficult, our results indicate that a controller can learn a robust peg insertion strategy.

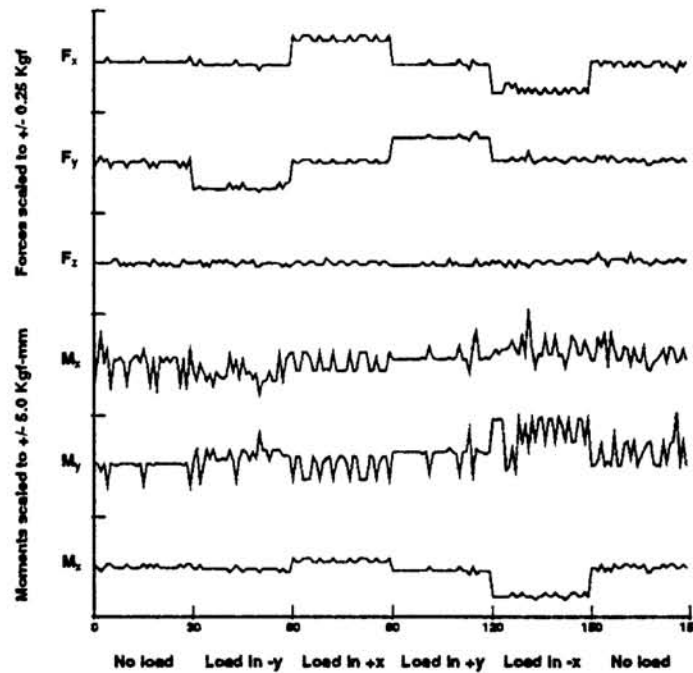

Figure 1: 30 Time-step Samples Of The Sensed Forces and Moments Under 5 Different Load Conditions. With An Ideal Sensor, The Readings Would Be Constant In Each 30 Time-step Interval.

## 4  LEARNING PEG-IN-HOLE INSERTION

Our approach to learning a reactive control strategy for peg insertion under uncertainty is based on active generation of compliant behavior using a nonlinear mapping from sensed positions and forces to position commands.[2] The controller learns this mapping through repeated attempts at peg insertion.

**The Peg Insertion Tasks**   As depicted in Figure 2, both 2D and 3D versions of the peg insertion task were attempted. In the 2D version of the task, the peg used was $50mm$ long and $22.225mm$ $(7/8in)$ wide, while the hole was $23.8125mm$ $(15/16in)$ wide. Thus the clearance between the peg and the hole was $0.79375mm$ $(1/32in)$. In the 3D version, the peg used was $30mm$ long and $6mm$ in diameter, while the hole was $6.35mm$ in diameter. Thus the clearance in the 3D case was $0.175mm$.

**The Controller**   The controller was implemented as a connectionist network that operated in closed loop with the robot so that it could learn a reactive control strategy for performing peg insertions. The network used in the 2D task had 6 inputs, viz., the sensed positions and forces, $(X, Y, \Theta)$ and $(F_x, F_y, M_z)$, three

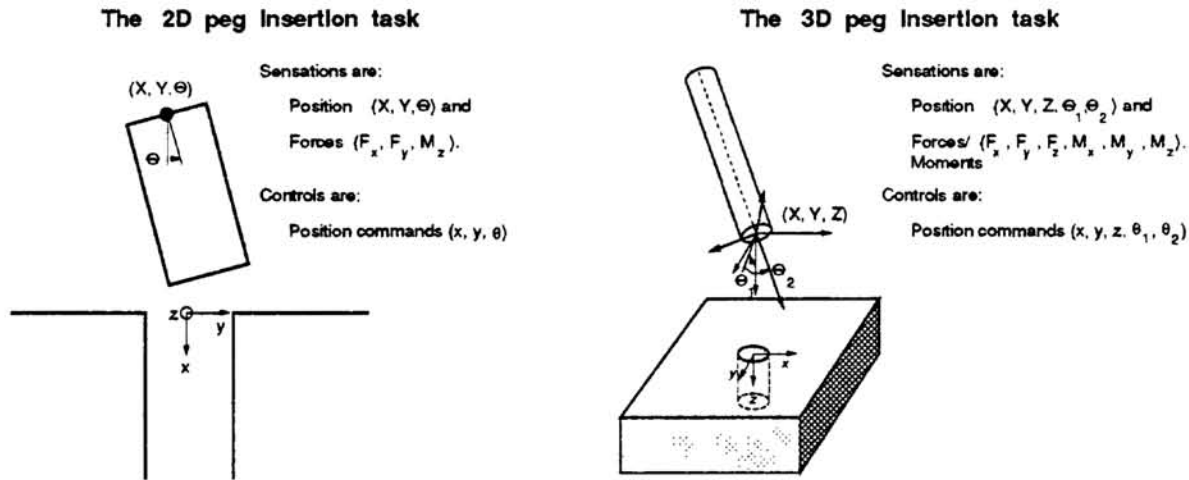

Figure 2: The 2D And 3D Peg-in-hole Insertion Tasks.

outputs forming the position command $(x, y, \theta)$, and two hidden layers of 15 units each. For the 3D task, the network had 11 inputs, the sensed positions and forces, $(X, Y, Z, \Theta_1, \Theta_2)$ and $(F_x, F_y, F_z, M_x, M_y, M_z)$, five outputs forming the position command $(x, y, z, \theta_1, \theta_2)$, and two hidden layers of 30 units each.

In both networks, the hidden units used were back-propagation units, while the output units used were stochastic real-valued (SRV) reinforcement learning units (Gullapalli, 1990). SRV units use a direct reinforcement learning algorithm to find the best real-valued output for each input (see Gullapalli (1990) for details). The position inputs to the network were computed from the sensed joint positions using the forward kinematics equations for the Zero. The force and moment inputs were those sensed by the six-axis force sensor. A PD servo loop was used to servo the robot to the position output by the network at each time step.

**Training Methodology**     The controller network was trained in a sequence of trials, each of which started with the peg at a random position and orientation with respect to the hole and ended either when the peg was successfully inserted in the hole, or when 100 time steps had elapsed. An insertion was termed successful when the peg was inserted to a depth of $25mm$ into the hole. At each time step during training, the sensed peg position and forces were input to the network, and the computed control output was executed by the robot, resulting in some motion of the peg. An evaluation of the controller's performance, $r$, ranging from 0 to 1 with 1 denoting the best possible evaluation, was computed based on the new peg position and the forces acting on the peg as

$$r = \begin{cases} \max(0.0, 1.0 - 0.01\|\text{position error}\|) & \text{if all forces} \leq 0.5\text{Kgf}, \\ \max(0.0, 1.0 - 0.01\|\text{position error}\| - 0.1F_{\max}) & \text{otherwise}, \end{cases}$$

where $F_{\max}$ denotes the largest magnitude force component. Thus, the closer the sensed peg position was to the desired position with the peg inserted in the hole, the higher the evaluation. Large sensed forces, however, reduced the evaluation. Using this evaluation, the network adjusted its weights appropriately and the cycle was repeated.

## 5    PERFORMANCE RESULTS

A learning curve showing the final evaluation over 500 consecutive trials on the 2D task is shown in Figure 3 (a). The final evaluation levels off close to 1 after about

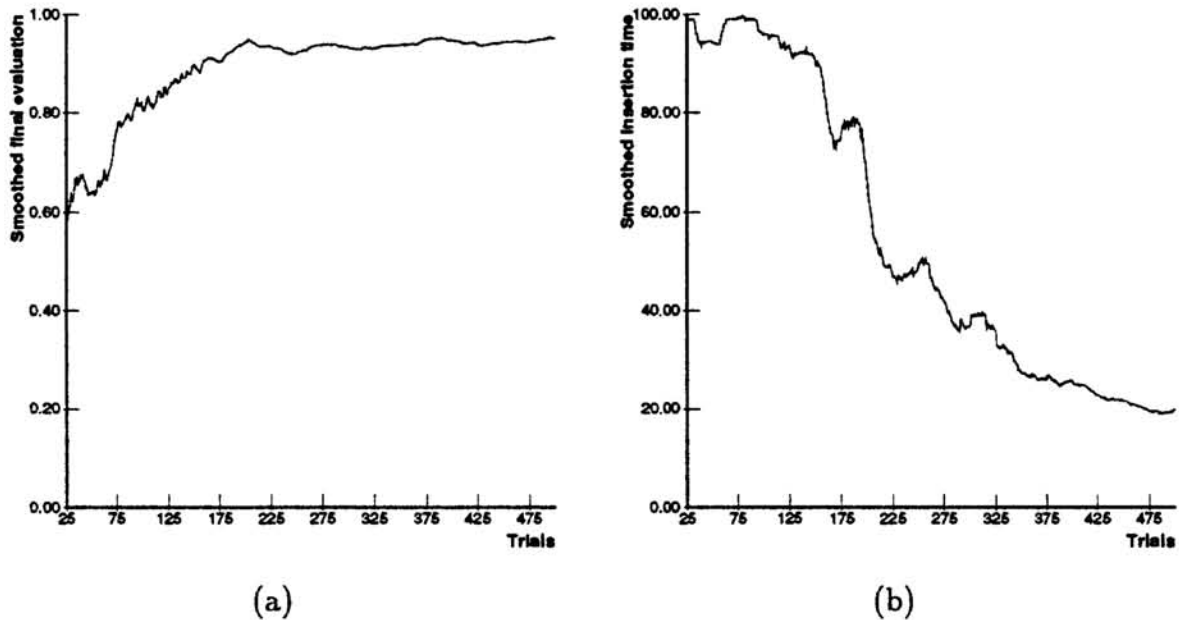

(a)                                                          (b)

Figure 3: Smoothed Final Evaluation Received And Smoothed Insertion Time (In Simulation Time Steps) Taken On Each Of 500 Consecutive Trials On The 2D Peg Insertion Task. The Smoothed Curve Was Obtained By Filtering The Raw Data Using A Moving-Average Window Of 25 Consecutive Values.

150 trials because after that amount of training, the controller is consistently able to perform successful insertions within 100 time steps. However, performance as measured by insertion time continues to improve, as is indicated by the learning curve in Figure 3 (b), which shows the time to insertion decreasing continuously over the 500 trials. These curves indicate that the controller becomes progressively more *skillful* at peg insertion with training. Similar results were obtained for the 3D task, although learning was slower in this case. The performance curves for the 3D task are shown in Figure 4.

## 6    DISCUSSION AND CONCLUSIONS

The high degree of uncertainty in the sensory feedback from the Zebra Zero, coupled with the fine motion control requirements of peg-in-hole insertion make the task under consideration an example of learning control under extreme uncertainty. The positional uncertainty, in particular, is of the order of 10 to 50 times the clearance between the peg and the hole and is primarily due to gear backlash. There is also significant uncertainty in the sensed forces and moments due to sensor noise. Our results indicate that direct reinforcement learning can be used to learn a reactive control strategy that works robustly even in the presence of a high degree of uncertainty.

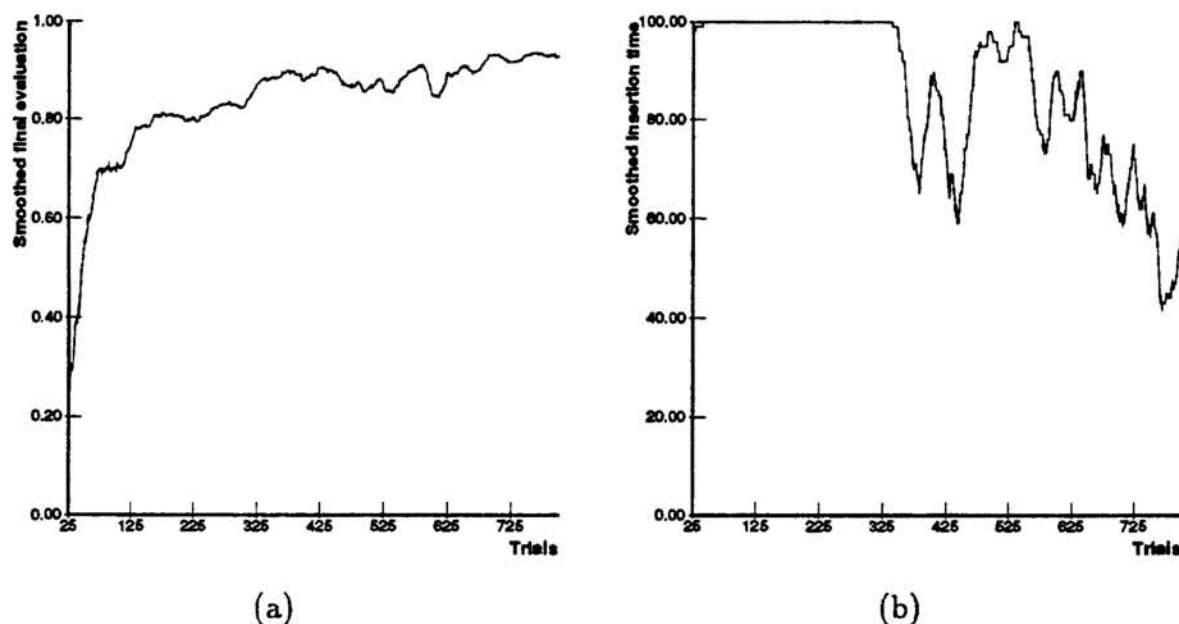

(a)                                                    (b)

Figure 4: Smoothed Final Evaluation Received And Smoothed Insertion Time (In Simulation Time Steps) Taken On Each Of 800 Consecutive Trials On The 3D Peg Insertion Task. The Smoothed Curve Was Obtained By Filtering The Raw Data Using A Moving-Average Window Of 25 Consecutive Values.

Although others have studied similar tasks, in most other work on learning peg-in-hole insertion (e.g., Lee & Kim, 1988) it is assumed that the positional uncertainty is about an order of magnitude *less* than the clearance. Moreover, results are often presented using simulated peg-hole systems. Our results indicate that our approach works well with a physical system, despite the much higher magnitudes of noise and consequently greater degree of uncertainty inherent in dealing with physical systems. Furthermore, the success of the direct reinforcement learning approach to training the controller indicates that this approach can be useful for automatically synthesizing robot control strategies that satisfy constraints encoded in the performance evaluations.

## Acknowledgements

This paper has benefited from many useful discussions with Andrew Barto and Roderic Grupen. I would also like to thank Kamal Souccar for assisting with running the Zebra Zero. This material is based upon work supported by the Air Force Office of Scientific Research, Bolling AFB, under Grant AFOSR-89-0526 and by the National Science Foundation under Grant ECS-8912623.

## Footnotes

[1] For our purposes, noise can be regarded simply as one of the sources of uncertainty.

[2]See also (Gullapalli et al., 1992).

## References

[1] H. Asada. Teaching and learning of compliance using neural nets: Representation and generation of nonlinear compliance. In *Proceedings of the 1990 IEEE International Conference on Robotics and Automation*, pages 1237–1244, 1990.

[2] A. G. Barto and P. Anandan. Pattern recognizing stochastic learning automata. *IEEE Transactions on Systems, Man, and Cybernetics*, 15:360–375, 1985.

[3] A. G. Barto and V. Gullapalli. Neural Networks and Adaptive Control. In P. Rudomin, M. A. Arbib, and F. Cervantes-Perez, editors, *Natural and Artificial Intelligence. Research Notes in Neural Computation*, Springer-Verlag: Washington. (in press).

[4] M. E. Caine, T. Lozano-Pérez, and W. P. Seering. Assembly strategies for chamferless parts. In *Proceedings of the IEEE International Conference on Robotics and Automation*, pages 472–477, May 1989.

[5] B. R. Donald. Robot motion planning with uncertainty in the geometric models of the robot and environment: A formal framework for error detection and recovery. In *Proceedings of the IEEE International Conference on Robotics and Automation*, pages 1588–1593, 1986.

[6] M. Erdmann. Using backprojections for fine motion planning with uncertainty. *International Journal of Robotics Research*, 5(1):19–45, 1986.

[7] S. J. Gordon. *Automated assembly using feature localization*. PhD thesis, Massachusetts Institute of Technology, MIT AI Laboratory, Cambridge, MA, 1986. Technical Report 932.

[8] V. Gullapalli. A stochastic reinforcement learning algorithm for learning real-valued functions. *Neural Networks*, 3:671–692, 1990.

[9] V. Gullapalli. *Reinforcement Learning and its application to control*. PhD thesis, University of Massachusetts, Amherst, MA 01003, 1992.

[10] V. Gullapalli, R. A. Grupen, and A. G. Barto. Learning reactive admittance control. In *Proceedings of the 1992 IEEE International Conference on Robotics and Automation*, pages 1475–1480, Nice, France, 1992.

[11] R. E. Gustavson. A theory for the three-dimensional mating of chamfered cylindrical parts. *Journal of Mechanisms, Transmissions, and Automated Design*, December 1984.

[12] S. Lee and M. H. Kim. Learning expert systems for robot fine motion control. In H. E. Stephanou, A. Meystal, and J. Y. S. Luh, editors, *Proceedings of the 1988 IEEE International Symposium on Intelligent Control*, pages 534–544, Arlington, Virginia, USA, 1989. IEEE Computer Society Press: Washington.

[13] T. Lozano-Pérez, M. T. Mason, and R. H. Taylor. Automatic synthesis of fine-motion strategies for robots. *The International Journal of Robotics Research*, 3(1):3–24, Spring 1984.

[14] D. E. Whitney. Quasi-static assembly of compliantly supported rigid parts. *Journal of Dynamic Systems, Measurement, and Control*, 104, March 1982. Also in *Robot Motion: Planning and Control*, (Brady, M., et al. eds.), MIT Press, Cambridge, MA, 1982.
